# Extended Regularization Methods for Nonconvergent Model Selection

**W. Finnoff, F. Hergert and H.G. Zimmermann**
Siemens AG, Corporate Research and Development
Otto-Hahn-Ring 6
8000 Munich 83, Fed. Rep. Germany

## Abstract

Many techniques for model selection in the field of neural networks correspond to well established statistical methods. The method of 'stopped training', on the other hand, in which an oversized network is trained until the error on a further validation set of examples deteriorates, then training is stopped, is a true innovation, since model selection doesn't require convergence of the training process.

In this paper we show that this performance can be significantly enhanced by extending the 'nonconvergent model selection method' of stopped training to include dynamic topology modifications (dynamic weight pruning) and modified complexity penalty term methods in which the weighting of the penalty term is adjusted during the training process.

## 1 INTRODUCTION

One of the central topics in the field of neural networks is that of model selection. Both the theoretical and practical side of this have been intensively investigated and a vast array of methods have been suggested to perform this task. A widely used class of techniques starts by choosing an 'oversized' network architecture then either removing redundant elements based on some measure of saliency (pruning), adding a further term to the cost function penalizing complexity (penalty terms), and finally, observing the error on a further validation set of examples, then stopping training as soon as this performance begins to deteriorate (stopped training). The first two methods can be viewed as variations of long established statistical techniques corresponding in the case of pruning to specification searches, and with respect to penalty terms as regularization or biased regression.

The method of stopped training, on the other hand, seems to be one of the true innovations to come out of neural network research. Here, the model chosen doesn't require the training process to converge, rather, the training process is used to perform a directed search of weight space to find a model with superior generalization performance. Recent theoretical ([B,C,91], [F,91], [F,Z,91]) and empirical results ([H,F,Z,92], [W,R,H,90]) have provided strong evidence for the efficiency of stopped training. In this paper we will show that generalization performance can be further enhanced by expanding the 'nonconvergent method' of stopped training to include dynamic topology modifications (dynamic pruning) and modified complexity penalty term methods in which the weighting of the penalty term is adjusted during the training process. Here, the empirical results are based on an extensive sequence of simulation examples designed to reduce the effects of domain dependence on the performance comparisons.

## 2 CLASSICAL MODEL SELECTION

Classical model selection methods are generally divided into a number of steps that are performed independently. The first step consists of choosing a network architecture, then either an objective function (possibly including a penalty term) is chosen directly, or in a Bayesian setting, prior distributions on the elements of the data generating process (noise, weights in the model, regularizers, etc.) are specified from which an objective function is derived. Next, using the specified objective function, the training process is started and continued until a convergence criterion is fulfilled. The resulting parametrization of the given architecture is then placed in a 'pool' from which a final model will be selected.

The next step can consist of a modification of the network architecture (for example by pruning weights/hidden-neurons/input-neurons), or of the penalty term (for example by changing its weighting in the objective function) or of the Bayesian prior distributions. The last two modifications then result in a modification of the objective function. This establishes a new framework for the training process which is then restarted and continued until convergence, producing another model for the pool. This process is iterated until the model builder is satisfied that the pool contains a reasonable diversity of candidate models, which are then compared with one another using some estimator of generalization ability, (for example, the performance on a validation set).

Stopped training, on the other hand, has a fundamentally different character. Although the choice of framework remains the same, the essential innovation consists of considering every parametrization of a given architecture as a potential model. This contrasts with classical methods in which only those parametrizations corresponding to minima of the objective function are taken into consideration for the model pool.

Under the weight of accumulated empirical evidence (see [W,R,H,90], [H,F,Z,92]) theorists have begun to investigate the properties of this technique and have been able to show that stopped training has the same sort of regularization effect (i.e. reduction of model variance at the cost of bias) that penalty terms provide (see

[B,C,91], [F,91]). Since the basic effect of pruning procedures is also to reduce network complexity (and consequent model variance) one sees that there is a close relationship in the instrumental effects of stopped training, pruning and regularization. The question remains whether (or under what circumstances) any one or combination of these methods produces superior results.

## 3    THE METHODS TESTED

In our expirements a single hidden layer feedforward network with tanh activation functions and ten hidden units was used to fit data sets generated in such a manner that network complexity had to be reduced or constrained to prevent overfitting. A variety of both classical and nonconvergent methods were tested for this purpose. The first we will discuss used weight pruning. To characterize the relevance of a weight in a given network, three different test variables were used. The first simply measures weight size under the assumption that the training process naturally forces nonrelevant weights into a region around zero. The second test variable is that used in the Optimal Brain Damage (OBD) pruning procedure of Le Cun et al. (see [L,D,S,90]). The final test variables considered are those proposed by Finnoff and Zimmermann in [F,Z,91], based on significance tests for deviations from zero in the weight update process.

Two pruning algorithms were used in the experiments, both of which attempt to emulate successful interactive methods. In the first algorithm, one removes a certain fixed percentage of weights in the network after a stopping criterion is reached. The reduced network is then trained further until the stopping criterion is once again fulfilled. This process is then repeated until performance breaks down completely. This method will be referred to in the following as *auto-pruning* and was implemented using all three types of test variables to determine the weights to be removed. The only difference lay in the stopping criterion used. In the case of the OBD test variables, training was stopped after the training process converged. In the case of the statistical and small weight test variables, training was stopped whenever overtraining (defined by a repeated increase in the error on a validation set) was observed. A final (restart) variant of auto-pruning using the statistical test variables was also tested. This version of auto-pruning only differs in that the weights are reinitialized (on the reduced topology) after every pruning step. In the tables of results presented in the appendix, the results for auto-pruning using the statistical (resp. small weight, resp. OBD) test variables will be denoted by P* (resp. G*, resp. O*). The version of auto-pruning using restarts will be denoted by p*.

The second method uses the statistical test variables to both remove and reactivate weights. As in auto-pruning the network is trained until overfitting is observed after a fixed number of epochs, then test values are calculated for all active and inactive weights. Here a fixed number $\varepsilon > 0$ is given, corresponding to some quantile value of a probability distribution. If the test variable for an active weight falls below $\varepsilon$ the weight is pruned (deactivated). For weights that have already been set to zero, the value of the test variables are compared with $\varepsilon$, and if larger, the weight is reactivated with a small random value. Furthermore, the value of $\varepsilon$ is increased by some $\Delta\varepsilon > 0$ after each pruning step until some value $\varepsilon_{max}$ is reached. This method is referred to as *epsi-pruning*. Epsi-pruning was tested in versions both with (e*)

and without restarts (E*).

Two complexity penalty terms were considered. These consist of a further term $C_\lambda(w)$ added to the error function which forces the network to achieve a compromise between fit and network complexity during the training process; here, the parameter $\lambda \in [0, \infty)$ controls the strength of the complexity penalty. The first is the quadratic term, the first derivative of which leads to the so-called weight decay term in the weight updates (see [H,P,89]). The second is the Weigend/Rumelhart penalty term (see [W,R,H,91]). The weight decay penalty term was tested using two techniques. In the first of these, (D*), $\lambda$ was held constant throughout the training process. In the second, (d*), $\lambda$ was set to zero until overtraining was observed, then turned on and held constant for the remainder of the training process. The Weigend/Rumelhart penalty term was also tested using these two methods (denoted in the following tables by W*, resp. w*). Further, the algorithm suggested by A. Weigend in [W,R,H,91] in which the value of $\lambda$ is varied during training was considered (wF).

In addition to the pruning and penalty term methods investigated, two (simple) versions of stopped training were tested, in one case (nN) with a constant learning step throughout, and in the other (nF) with the step size reduced after overtraining was observed. Finally three benchmarks were included. All these involved training a network until convergence to emulate the situation when no precautions are taken to prevent overfitting other than varying the number of hidden units. The number of hidden units in these benchmark tests was set at three, six and ten, (#3, #6, ##) this last network having then the same topology as that used in the remaining tests.

## 4    THE DATA GENERATION PROCESSES

To test the methods under consideration, a number of processes were used to generate data sets. By testing on a sufficiently wide range of controlled examples one hopes to reduce the domain dependence that might arise in the performance comparisons. The data used in our experiments was based on pairs $(\tilde{y}_i, x_i)$, $i = 1, ..., T$, $T \in \mathbf{N}$ with targets $\tilde{y}_i \in \mathbf{R}$ and inputs $x_i = (x_i^1, ..., x_i^K) \in [-1, 1]^K$, where $\tilde{y}_i = g(x_i^1, ..., x_i^j) + u_i$, for $j, K \in \mathbf{N}$. Here, $g$ represents the *structure* in the data, $x^1, ..., x^j$ the *relevant* inputs, $x^{j+1}, ..., x^K$, the irrelevant or *decoy* inputs and $u_i$ a stochastic disturbance term.

The first group of experiments was based on an additive structure $g$ having the following form with $j = 5$ and $K = 10$, $g(x_i^1, ..., x_i^5) = \sum_{k=1}^{5} f(\alpha^k x_i^k)$, $\alpha^k \in \mathbf{R}$ and $f$ either the identity on $\mathbf{R}$ or sin. The second class of models investigated had a highly nonlinear product structure $g$ with $j = 3$, $K = 10$ and $g(x_i^1, ..., x_i^3) = \prod_{k=1}^{3} f(\alpha^k x_i^k)$, $\alpha^k \in \mathbf{R}$ and $f$ once again either the identity on $\mathbf{R}$ or sin. The next structure considered was constructed using sums of Radial Basis Functions (RBF's) as follows, $g(x_i^1, ..., x_i^5) = \sum_{l=1}^{8} (-1)^l \exp\left(\sum_{k=1}^{5} \frac{(\alpha^{k,l} - x_i^k)^2}{2\sigma^2}\right)$, with $\alpha^{k,l} \in \mathbf{R}$ for $k = 1, ..., 5$, $l = 1, ..., 8$. Here, for every $l = 1, ..., 8$ the vector parameter $(\alpha^{1,l}, ..., \alpha^{5,l})$ corresponds to the center of the RBF. The final group of experiments were conducted using data generated by feedforward network activation functions. The network used for this task had fifty input units, two hundred hidden units and

one output. In every experiment, the data was divided into three disjoint subsets $\mathcal{D}_t, \mathcal{D}_v, \mathcal{D}_g$: The first set $\mathcal{D}_t$ was used for training, the second $\mathcal{D}_v$ (validation) set to test for overfitting and to steer the pruning algorithms and the third $\mathcal{D}_g$ (generalization) set to test the quality of the model selection process.

## 5    DISCUSSION

The results of the experiments are given below. Here we give a short review of the most interesting phanomena observed.

Notable in a general sense is a striking domain dependence in the performance, which illustrates the danger of basing a comparison of methods on tests using a single (particularly small) data set. Another valuable observation is that by testing at higher levels of significance, apparent performance differences can dwindle or even disappear. Finally, one sees that even in the examples without noise that overfitting occurs, which contradicts the frequently stated conviction that overfitting is noise fitting.

With regard to specific methods, one sees that all the methods tested significantly improved generalization performance when compared to the benchmarks. Further, the results show that the extended nonconvergent methods are on average superior (sometimes dramatically so) than their classical counterparts. In particular, the performance of penalty terms is greatly enhanced if they are first introduced in the training process *after* overtraining is observed. Further, dynamic pruning using the statistical or even the small weight test variables produces significantly better results than stopped training alone or using the Optimal Brain Damage (OBD) weight elimination method which requires training to minima of the objective function. A final notable observation is that the pruning methods (especially those using resarts) generally work better in the examples with a great deal of noise, while the penalty term methods are superior when the structure is highly nonlinear.

## 6    TABLES OF RESULTS

The experiments were performed as follows: First, each data generating process was used to produce six independent sets of data and initial weights to increase the statistical significance of observed effects and to help reduce the effects of any data set specific anomalies. In a second step, the parameters of the training processes were optimized for each example by extensive testing, then a fixed value for each parameter was chosen for use across the entire range of experiments. With these parameters, each method was tested on all of the six data sets produced by one data generating process. Both the penalty terms and the pruning methods were tested with different settings of the relevant parameters in each model. The parameter values used in the simulations and an overview of the methods tested are collected in the following two tables.

## 6.1    Parameter Settings of the Experiments

| Symbol | Structure | Noise Var. | Size $\mathcal{D}_t/\mathcal{D}_v/\mathcal{D}_g$ | Learn Step before/after overfitting |
|---|---|---|---|---|
| exp_0_n | Sum of RBF's | 0.0 | 400/200/1000 | 0.05/0.005 |
| exp_3_n | Sum of RBF's | 0.3 | 400/200/1000 | 0.05/0.005 |
| exp_6_n | Sum of RBF's | 0.6 | 400/200/1000 | 0.05/0.005 |
| id_7_n | $\sum_{k=0}^{5} \alpha_k x_k$ | 0.7 | 200/100/1000 | 0.05/0.005 |
| id_8_n | $\sum_{k=0}^{5} \alpha_k x_k$ | 0.8 | 200/100/1000 | 0.05/0.005 |
| id_9_n | $\sum_{k=0}^{5} \alpha_k x_k$ | 0.9 | 200/100/1000 | 0.05/0.005 |
| n_0_id | $\prod_{k=0}^{3} x_k$ | 0.0 | 1400/600/1000 | 0.05/0.01 |
| n_1_id | $\prod_{k=0}^{3} x_k$ | 0.1 | 1400/600/1000 | 0.05/0.01 |
| n_2_id | $\prod_{k=0}^{3} x_k$ | 0.2 | 1400/600/1000 | 0.05/0.01 |
| n_0_sin | $\sum_{k=0}^{5} \sin(\alpha_k x_k)$ | 0.0 | 1400/600/1000 | 0.05/0.01 |
| n_1_sin | $\sum_{k=0}^{5} \sin(\alpha_k x_k)$ | 0.1 | 1400/600/1000 | 0.05/0.01 |
| n_2_sin | $\sum_{k=0}^{5} \sin(\alpha_k x_k)$ | 0.2 | 1400/600/1000 | 0.05/0.01 |
| net_0_n | 50/200/1 Network | 0.0 | 400/200/1000 | 0.05/0.005 |
| net_3_n | 50/200/1 Network | 0.3 | 400/200/1000 | 0.05/0.005 |
| net_6_n | 50/200/1 Network | 0.6 | 400/200/1000 | 0.05/0.005 |
| sin_0_n | $\sum_{k=0}^{5} \sin(\alpha_k x_k)$ | 0.0 | 400/200/1000 | 0.05/0.005 |
| sin_3_n | $\sum_{k=0}^{5} \sin(\alpha_k x_k)$ | 0.3 | 400/200/1000 | 0.05/0.005 |
| sin_6_n | $\sum_{k=0}^{5} \sin(\alpha_k x_k)$ | 0.6 | 400/200/1000 | 0.05/0.005 |

## 6.2    Overview of Methods Tested

| Method Type | Method | Explanation |
|---|---|---|
| Classical | D* | Weight Decay, constant $\lambda$ |
| Penalty Term | W* | Weigend/Rumelhart, constant $\lambda$ |
| Nonconvergent | d* | Weight Decay, 1-Step $\lambda$ adaptation |
| Penalty Term | w* | Weigend/Rumelhart, 1-Step $\lambda$ adaptation |
| | wF | Weigend/Rumelhart, $\lambda$-adapt by Weigend algorithm |
| Auto-Pruning | G* | Small weight pruning |
| | o* | Optimal Brain Damage |
| | P* | Pruning with statistical test var., no restarts |
| | p* | Pruning with statistical test var., using restarts |
| Epsi-Pruning | E* | Pruning with statistical test var., no restarts |
| | e* | Pruning with statistical test var., using restarts |
| Stopped | nF | Stopped training with reduced $\eta$ |
| Training | nN | Stopped training with constant $\eta$ |
| Benchmarks | 3# | Training to convergence, 3 Neurons im hidden Layer |
| | 6# | Training to convergence, 6 neurons im hidden layer |
| | ## | Training to convergence, 10 neurons im hidden layer |

The following tables give categorical rankings of the results. The rankings were calculated as follows: The method with the best performance was given ranking 1, then the performance of each following method was compared with that of the method on the first position using a modified $t$-test statistic. The first method in the list whose test results deviated from that on the first position to at least the quantile value of the statistic given at the head of the table was then used to start the second category. All those whose test results did not deviate by at least this amount were given the same ranking as the leading method of the category, (in this case 1). Following categories were then formed in an analogous fashion using test results measured against the performance of the leading method at the head of the category.

The results are presented in two tables. The first contains the results for the data generating processes without noise and the second for the models with noise. The categorical rankings given were determined using the procedure described above at a 0.9 level of significance. The ordering of the methods given, listed in the first column, is based on the average ranking over all the simulations listed in the table. This average is given in the second column.

### 6.2.1   Data Generating Processes without Noise

Classification by objective function, $t_\alpha = 0.9$

| method | av | exp_0_n | n_0_id | n_0_sin | net_0_n | sin_0_n |
|--------|-----|---------|--------|---------|---------|---------|
| d*     | 1.6 | 1       | 3      | 1       | 2       | 1       |
| P*     | 1.8 | 2       | 2      | 2       | 1       | 2       |
| w*     | 2.0 | 1       | 2      | 3       | 3       | 1       |
| wF     | 2.0 | 2       | 1      | 3       | 2       | 2       |
| G*     | 2.2 | 2       | 3      | 3       | 1       | 2       |
| E*     | 2.6 | 2       | 5      | 3       | 1       | 2       |
| o*     | 2.6 | 3       | 4      | 2       | 2       | 2       |
| p*     | 2.6 | 4       | 5      | 2       | 1       | 1       |
| nF     | 3.0 | 3       | 5      | 3       | 2       | 2       |
| e*     | 3.8 | 4       | 6      | 4       | 3       | 2       |
| nN     | 3.8 | 4       | 7      | 4       | 1       | 3       |
| ##     | 5.2 | 5       | 8      | 4       | 3       | 6       |
| W*     | 5.6 | 8       | 10     | 7       | 1       | 2       |
| D*     | 5.8 | 8       | 11     | 7       | 2       | 1       |
| 6#     | 6.2 | 6       | 9      | 6       | 5       | 5       |
| 3#     | 6.4 | 7       | 12     | 5       | 4       | 4       |

### 6.2.2  Data Generating Processes with Noise

Classification by objective function, $t_\alpha = 0.9$

| method | av | exp _3_n | exp _6_n | id _9_n | n_1 _id | n_1 _sin | n_2 _id | n_2 _sin | net _3_n | net _6_n | sin _3_n | sin _6_n |
|---|---|---|---|---|---|---|---|---|---|---|---|---|
| P* | 2.1 | 3 | 1 | 4 | 2 | 2 | 1 | 3 | 3 | 1 | 2 | 2 |
| d* | 2.2 | 5 | 5 | 3 | 1 | 1 | 2 | 1 | 1 | 1 | 2 | 2 |
| p* | 2.2 | 2 | 1 | 1 | 3 | 5 | 3 | 2 | 1 | 2 | 1 | 1 |
| wF | 2.2 | 4 | 5 | 3 | 1 | 4 | 1 | 2 | 2 | 1 | 2 | 1 |
| e* | 2.2 | 1 | 2 | 1 | 4 | 5 | 3 | 4 | 2 | 2 | 1 | 1 |
| E* | 2.6 | 3 | 3 | 2 | 4 | 4 | 3 | 3 | 2 | 1 | 2 | 2 |
| G* | 2.7 | 4 | 4 | 3 | 3 | 3 | 3 | 3 | 2 | 1 | 2 | 2 |
| o* | 2.8 | 5 | 5 | 3 | 3 | 5 | 4 | 1 | 1 | 1 | 2 | 1 |
| w* | 2.8 | 5 | 5 | 3 | 3 | 5 | 3 | 3 | 1 | 1 | 2 | 2 |
| nF | 2.9 | 5 | 5 | 3 | 3 | 5 | 3 | 3 | 1 | 1 | 2 | 2 |
| nN | 3.5 | 5 | 5 | 4 | 4 | 5 | 3 | 3 | 3 | 1 | 3 | 3 |
| D* | 3.7 | 5 | 4 | 5 | 1 | 5 | 7 | 5 | 2 | 2 | 1 | 1 |
| W* | 4.1 | 5 | 4 | 5 | 5 | 6 | 7 | 5 | 2 | 2 | 1 | 1 |
| ## | 5.2 | 6 | 6 | 6 | 6 | 5 | 5 | 5 | 5 | 4 | 5 | 5 |
| 3# | 5.3 | 7 | 7 | 7 | 7 | 5 | 7 | 6 | 4 | 3 | 4 | 4 |
| 6# | 5.4 | 8 | 8 | 8 | 5 | 4 | 6 | 3 | 4 | 4 | 5 | 5 |

## 7   REFERENCES

[B,C,91] Baldi, P. and Chauvin, Y., Temporal evolution of generalization during learning in linear networks, *Neural Computation* 3, 1991, pp. 589-603.

[F,91] Finnoff, W., Complexity measures for classes of neural networks with variable weight bounds, in *Proc. Int. Joint Conf. on Neural Networks*, Singapore, 1991.

[F,Z,91] Finnoff, W., Zimmermann, H.G., Detecting structure in small datasets by network fitting under complexity constraints, to appear in *Proc. of 2nd Ann. Workshop Computational Learning Theory and Natural Learning Systems*, Berkeley, 1991.

[H,P,89], Hanson, S. J., and Pratt, L. Y., Comparing biases for minimal network construction with back-propagation, in *Advances in Neural Information Processing I*, D. S. Touretzky, Ed., Morgan Kaufman, 1989.

[H,F,Z,92] Hergert, F., Finnoff, W. and H.G. Zimmermann, A comparison of weight elimination methods for reducing complexity in neural networks. To be presented at *Int. Joint Conf. on Neural Networks*, Baltimore, 1992.

[L,D,S,90] Le Cun, Y., Denker J. and Solla, S., Optimal Brain Damage, in *Proceedings of Neural Information Processing Systems II*, Denver, 1990.

[W,R,H,91] Weigend, A., Rumelhart, D., and Huberman, B., Generalization by weight elimination with application to forecasting, *Advances in Neural Information Processing III*, Ed. R. P. Lippman and J. Moody, Morgan Kaufman, 1991.
